# Learning Path Distributions using Nonequilibrium Diffusion Networks

**Paul Mineiro** *
pmineiro@cogsci.ucsd.edu
Department of Cognitive Science
University of California, San Diego
La Jolla, CA 92093-0515

**Javier Movellan**
movellan@cogsci.ucsd.edu
Department of Cognitive Science
University of California, San Diego
La Jolla, CA 92093-0515

**Ruth J. Williams**
williams@math.ucsd.edu
Department of Mathematics
University of California, San Diego
La Jolla, CA 92093-0112

## Abstract

We propose diffusion networks, a type of recurrent neural network with probabilistic dynamics, as models for learning natural signals that are continuous in time and space. We give a formula for the gradient of the log-likelihood of a path with respect to the drift parameters for a diffusion network. This gradient can be used to optimize diffusion networks in the nonequilibrium regime for a wide variety of problems paralleling techniques which have succeeded in engineering fields such as system identification, state estimation and signal filtering. An aspect of this work which is of particular interest to computational neuroscience and hardware design is that with a suitable choice of activation function, e.g., quasi-linear sigmoidal, the gradient formula is local in space and time.

## 1 Introduction

Many natural signals, like pixel gray-levels, line orientations, object position, velocity and shape parameters, are well described as continuous–time continuous–valued stochastic processes; however, the neural network literature has seldom explored the continuous stochastic case. Since the solutions to many decision theoretic problems of interest are naturally formulated using probability distributions, it is desirable to have a flexible framework for approximating probability distributions on continuous path spaces. Such a framework could prove as useful for problems involving continuous–time continuous–valued processes as conventional hidden Markov models have proven for problems involving discrete–time sequences.

Diffusion networks are similar to recurrent neural networks, but have probabilistic dynamics. Instead of a set of ordinary differential equations (ODEs), diffusion networks are described by a set of stochastic differential equations (SDEs). SDEs provide a rich language for expressing stochastic temporal dynamics and have proven

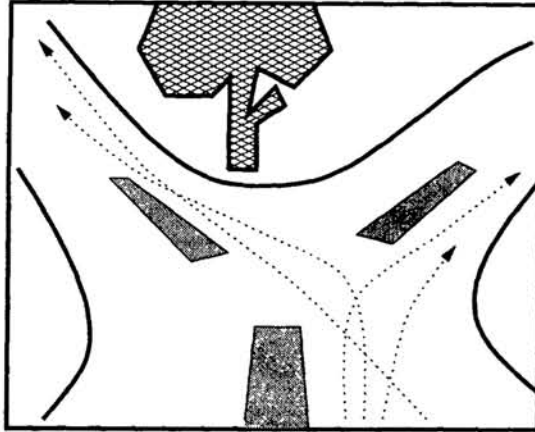

Figure 1: An example where the average of desirable paths yields an undesirable path, namely one that collides with the tree.

useful in formulating continuous–time statistical inference problems, resulting in such solutions as the continuous Kalman filter and generalizations of it like the condensation algorithm (Isard & Blake, 1996).

A formula is given here for the gradient of the log-likelihood of a path with respect to the drift parameters for a diffusion network. Using this gradient we can potentially optimize the model to approximate an entire probability distribution of continuous paths, not just average paths or equilibrium points. Figure 1 illustrates the importance of this kind of learning by showing a case in which learning average paths would have undesirable results, namely collision with a tree. Experience has shown that learning distributions of paths, not just averages, is crucial for dynamic perceptual tasks in realistic environments, e.g., visual contour tracking (Isard & Blake, 1996). Interestingly, with a suitable choice of activation function, e.g., quasi-linear sigmoidal, the gradient formula depends only upon local computations, i.e., no time unfolding or explicit backpropagation of error is needed. The fact that noise localizes the gradient is of potential interest for domains such as theoretical neuroscience, cognitive modeling and hardware design.

## 2 Diffusion Networks

Hereafter $C_n$ refers to the space of continuous $\mathbf{R}^n$-valued functions over the time interval $[0, T]$, with $T \in \mathbf{R}$, $T > 0$ fixed throughout this discussion.

A diffusion network with parameter $\lambda \in \mathbf{R}^p$ is a random process defined via an Itô SDE of the form

$$dX(t) = \mu(t, X(t), \lambda)dt + \sigma dB(t), \tag{1}$$
$$X(0) \sim \nu,$$

where $X$ is a $C_n$-valued process that represents the temporal dynamics of the $n$ nodes in the network; $\mu : [0, T] \times \mathbf{R}^n \times \mathbf{R}^p \to \mathbf{R}^n$ is a deterministic function called the drift; $\lambda \in \mathbf{R}^p$ is the vector of drift parameters, e.g., synaptic weights, which are to be optimized; $B$ is a Brownian motion process which provides the random driving term for the dynamics; $\nu$ is the initial distribution of the solution; and $\sigma \in \mathbf{R}$, $\sigma > 0$, is a *fixed* constant called the dispersion coefficient, which determines the strength of the noise term. In this paper we do not address the problem of optimizing the dispersion or the initial distribution of $X$. For the existence and uniqueness in law of the solution to (1) $\mu(\cdot, \cdot, \lambda)$ must satisfy some conditions. For example, it is sufficient that it is Borel measurable and satisfies a linear growth condition: $|\mu(t, x, \lambda)| \leq K_\lambda(1 + |x|)$ for some $K_\lambda > 0$ and all $t \in [0, T]$, $x \in \mathbf{R}^n$; see

(Karatzas & Shreve, 1991, page 303) for details.

It is typically the case that the $n$-dimensional diffusion network will be used to model $d$-dimensional observations with $n > d$. In this case we divide $X$ into hidden and observable[1] components, denoted $H$ and $O$ respectively, so that $X = (H, O)$.

Note that with $\sigma = 0$ in equation (1), the model becomes equivalent to a continuous–time deterministic recurrent neural network. Diffusion networks can therefore be thought of as neural networks with "synaptic noise" represented by a Brownian motion process. In addition, diffusion networks have Markovian dynamics, and hidden states if $n > d$; therefore, they are also continuous–time continuous–state hidden Markov models. As with conventional hidden Markov models, the probability density of an observable state sequence plays an important role in the optimization of diffusion networks. However, because $X$ is a continuous–time process, care must taken in defining a probability density.

## 2.1  Density of a continuous observable path

Let $(X^\lambda, B^\lambda)$ defined on some filtered probability space $(\tilde{\Omega}, \tilde{F}, \{\tilde{F}_t\}, \tilde{P})$ be a (weak) solution of (1) with fixed parameter $\lambda$. Here $X^\lambda = (H^\lambda, O^\lambda)$ represents the states of the network and is adapted to the filtration $\{\tilde{F}_t\}$, $B^\lambda$ is an $n$-dimensional $\{\tilde{F}_t\}$-martingale Brownian motion and the filtration $\{\tilde{F}_t\}$ satisfies the usual conditions (Karatzas and Shreve, 1991, page 300). Let $Q^\lambda$ be the unique probability law generated by any weak solution of (1) with fixed parameter $\lambda$

$$Q^\lambda(A) = \tilde{P}(X^\lambda \in A) \text{ for all } A \in \mathcal{F}, \tag{2}$$

where $\mathcal{F}$ is the Borel sigma algebra generated by the open sets of $C_n$. Setting $\Omega = C_n$, $\Omega_h = C_{n-d}$, and $\Omega_o = C_d$ with associated Borel $\sigma$-algebras $\mathcal{F}$, $\mathcal{F}_h$ and $\mathcal{F}_o$, respectively, we have $\Omega = \Omega_h \times \Omega_o$, $\mathcal{F} = \mathcal{F}_h \otimes \mathcal{F}_o$, and we can define the marginal laws for the hidden and observable components of the network by

$$Q_h^\lambda(A_h) = Q^\lambda(A_h \times C_d) \triangleq \tilde{P}(H^\lambda \in A_h) \text{ for all } A_h \in \mathcal{F}_h, \tag{3}$$

$$Q_o^\lambda(A_o) = Q^\lambda(C_{n-d} \times A_o) \triangleq \tilde{P}(O^\lambda \in A_o) \text{ for all } A_o \in \mathcal{F}_o. \tag{4}$$

For our purposes the appropriate generalization of the notion of a probability density on $\mathbf{R}^m$ to the general probability spaces considered here is the Radon-Nikodym derivative with respect to a reference measure that dominates all members of the family $\{Q^\lambda\}_{\lambda \in \mathbf{R}^p}$ (Poor, 1994, p.264ff). A suitable reference measure $P$ is the law of the solution to (1) with zero drift ($\mu = 0$). The measures induced by this reference measure over $\mathcal{F}_h$ and $\mathcal{F}_o$ are denoted by $P_h$ and $P_o$, respectively. Since in the reference model there are no couplings between any of the nodes in the network, the hidden and observable processes are independent and it follows that

$$P(A_h \times A_o) = P_h(A_h)P_o(A_o) \text{ for all } A_h \in \mathcal{F}_h, A_o \in \mathcal{F}_o. \tag{5}$$

The conditions on $\mu$ mentioned above are sufficient to ensure a Radon-Nikodym derivative for each $Q^\lambda$ with respect to the reference measure. Using Girsanov's Theorem (Karatzas & Shreve, 1991, p.190ff) its form can be shown to be

$$Z^\lambda(\omega) = \frac{dQ^\lambda}{dP}(\omega) = \exp\left\{ \frac{1}{\sigma^2} \int_0^T \mu(t, \omega(t), \lambda) \cdot d\omega(t) \right.$$
$$\left. - \frac{1}{2\sigma^2} \int_0^T |\mu(t, \omega(t), \lambda)|^2 dt \right\}, \ \omega \in \Omega, \tag{6}$$

where the first integral is an Itô stochastic integral. The random variable $Z^\lambda$ can be interpreted as a likelihood or probability density with respect to the reference model[2]. However equation (6) defines the density of $\mathbf{R}^n$-valued paths of the entire network, whereas our real concern is the density of $\mathbf{R}^d$-valued observable paths. Denoting $\omega \in \Omega$ as $\omega = (\omega_h, \omega_o)$ where $\omega_h \in \Omega_h$ and $\omega_o \in \Omega_o$, note that

$$Q_o^\lambda(A) = \int_{\Omega_h \times \Omega_o} 1_A(\omega_o)\, Q^\lambda(d(\omega_h, \omega_o)) \tag{7}$$

$$= \int_{\Omega_o} 1_A(\omega_o) \left( \int_{\Omega_h} P_h(d\omega_h) Z^\lambda(\omega_h, \omega_o) \right) P_o(d\omega_o), \tag{8}$$

and therefore the Radon-Nikodym derivative of $Q_o^\lambda$ with respect to $P_o$, the density of interest, is given by

$$Z_o^\lambda(\omega_o) = \frac{dQ_o^\lambda}{dP_o}(\omega_o) = E^{P_h}[Z^\lambda(\cdot, \omega_o)], \quad \omega_o \in \Omega_o. \tag{9}$$

## 2.2 Gradient of the density of an observable path

The gradient of $Z_o^\lambda$ with respect to $\lambda$ is an important quantity for iterative optimization of cost functionals corresponding to a variety of problems of interest, e.g., maximum likelihood estimation of diffusion parameters for continuous path density estimation. Formal differentiation[3] of (9) yields

$$\nabla_\lambda \log Z_o^\lambda(\omega_o) = E^{P_h}[Z_{h|o}^\lambda(\cdot, \omega_o) \nabla_\lambda \log Z^\lambda(\cdot, \omega_o)], \tag{10}$$

where

$$Z_{h|o}^\lambda(\omega) \triangleq \frac{Z^\lambda(\omega)}{Z_o^\lambda(\omega_o)}, \tag{11}$$

$$\nabla_\lambda \log Z^\lambda(\omega) = \frac{1}{\sigma^2} \int_0^T J(t, \omega(t), \lambda) \cdot dI(\omega, t), \tag{12}$$

$$J_{jk}(t, x, \lambda) \triangleq \frac{\partial \mu_k(t, x, \lambda)}{\partial \lambda_j}, \tag{13}$$

$$I(\omega, t) \triangleq \omega(t) - \omega(0) - \int_0^t \mu(s, \omega(s), \lambda) ds. \tag{14}$$

Equation (10) states that the gradient of the density of an observable path can be found by clamping the observable nodes to that path and performing an average of $Z_{h|o}^\lambda \nabla_\lambda \log Z^\lambda$ with respect to $P_h$, i.e., average with respect to the hidden paths distributed as a scaled Brownian motion. This makes intuitive sense: the output gradient of the log density is a weighted average of the total gradient of the log density, where each hidden path contributes according to its likelihood $Z_{h|o}^\lambda$ given the output.

In practice to evaluate the gradient, equation (10) must be approximated. Here we use Monte Carlo techniques, the efficiency of which can be improved by sampling according to a density which reduces the variance of the integrand. Such a density

is available for models with hidden dynamics which do not explicitly depend upon the observables, i.e., the observable nodes do not send feedback connections to the hidden states. Models which obey this constraint are henceforth denoted *factorial*. Denoting $\mu_h$ and $\mu_o$ as the hidden and observable components, respectively, of the drift vector, and $B_h$ and $B_o$ as the hidden and observable components, respectively, of the Brownian motion, for a factorial network we have

$$dH(t) = \mu_h(t, H(t), \lambda)dt + \sigma dB_h(t), \tag{15}$$
$$dO(t) = \mu_o(t, H(t), O(t), \lambda)dt + \sigma dB_o(t). \tag{16}$$

The drift for the hidden variables does not depend on the observables, and Girsanov's theorem gives us an explicit formula for the density of the hidden process.

$$Z_h^\lambda(\omega_h) = \frac{dQ_h^\lambda}{dP_h}(\omega_h) = \exp\left\{ \frac{1}{\sigma^2} \int_0^T \mu_h(t, \omega_h(t), \lambda) \cdot d\omega_h(t) \right.$$
$$\left. -\frac{1}{2\sigma^2} \int_0^T |\mu_h(t, \omega_h(t), \lambda)|^2 dt \right\}. \tag{17}$$

Equations (9) and (10) can then be written in the form

$$Z_o^\lambda(\omega_o) = E^{Q_h^\lambda}[Z_{o|h}(\cdot, \omega_o)], \tag{18}$$

$$\nabla_\lambda \log Z_o^\lambda(\omega_o) = E^{Q_h^\lambda} \left[ \frac{Z_{o|h}^\lambda(\cdot, \omega_o)}{Z_o^\lambda(\omega_o)} \nabla_\lambda \log Z^\lambda(\cdot, \omega_o) \right], \tag{19}$$

where

$$Z_{o|h}^\lambda(\omega) \triangleq \frac{Z^\lambda(\omega)}{Z_h^\lambda(\omega_h)} = \exp\left\{ \frac{1}{\sigma^2} \int_0^T \mu_o(t, \omega(t), \lambda) \cdot d\omega_o(t) \right.$$
$$\left. -\frac{1}{2\sigma^2} \int_0^T |\mu_o(t, \omega(t), \lambda)|^2 dt \right\}. \tag{20}$$

Note the expectations are now performed using the measure $Q_h^\lambda$. We can easily generate samples according to $Q_h^\lambda$ by numerically integrating equation (15), and in practice this leads to more efficient Monte Carlo approximations of the likelihood and gradient.

## 3   Example: Noisy Sinusoidal Detection

This problem is a simple example of using diffusion networks for signal detection. The task was to detect a sinusoid in the presence of additive Gaussian noise. Stimuli were generated according to the following process

$$Y(t, \omega) = 1_A(\omega)\frac{1}{\pi}\sin(4\pi t) + B(t, \omega), \tag{21}$$

where $t \in [0, 1/2]$. Here $Y$ is assumed anchored in a probability space $(\Omega, \mathcal{F}, P)$ large enough to accommodate the event $A$ which indicates a signal or noise trial. Note that under $P$, $B$ is a Brownian motion on $C_d$ independent of $A$.

A model was optimized using 100 samples of equation (21) given $\omega \in A$, i.e., 100 stimuli containing a signal. The model had four hidden units and one observable unit ($n = 5$, $d = 1$). The drift of the model was given by

$$\mu(t, x, \lambda) = \theta + W \cdot g(x), \tag{22}$$

$$g_j(x) = \frac{1}{1 + e^{-x_j}}, \ j \in \{1, 2, 3, 4, 5\},$$

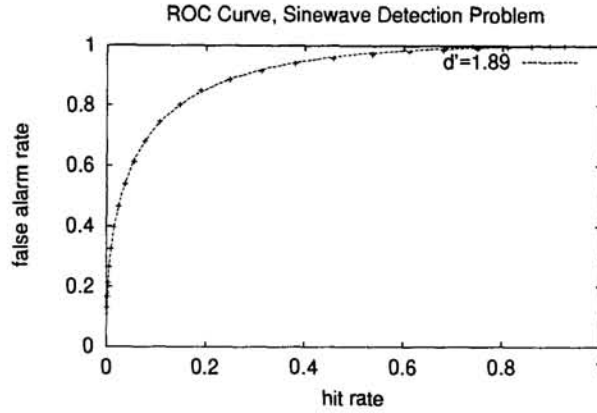

Figure 2: Receiver operating characteristic (ROC) curve for a diffusion network performing a signal detection task involving noisy sinusoids. Dotted line: Detection performance estimated numerically using 10000 novel stimuli. Solid line: Best fit curve corresponding to $d' = 1.89$. This value of $d'$ corresponds to performance within 1.5% of the Bayesian limit.

where $\theta \in \mathbf{R}^5$ and $W$ is a 5x5 real-valued connection matrix. In this case $\lambda = \{\{\theta_i\}, \{W_{ij}\}, i, j = 1, \dots, 5\}$. The connections from output to hidden units were set to zero, allowing use of the more efficient techniques for factorial networks described above. The initial distribution for the model was a $\delta$-function at $(1, -1, 1, -1, 0)$. The model was numerically simulated with $\Delta t = 0.01$, and 100 hidden samples were used to approximate the likelihood and gradient of the log–likelihood, according to equations (18) and (19). The conjugate gradient algorithm was used for training, with the log-likelihood of the data as the cost function.

Once training was complete, the parameter estimation was tested using 10000 novel stimuli and the following procedure. Given a new stimuli $y$ we used the model to estimate the likelihood $\hat{Z}_o(Y \mid A) \triangleq Z_o^{\hat{\lambda}}(Y)$, where $\hat{\lambda}$ is the parameter vector at the end of training. The decision rule employed was

$$D(Y) = \begin{cases} \text{signal} & \text{if } \hat{Z}_o(Y \mid A) > b, \\ \text{noise} & \text{otherwise,} \end{cases} \tag{23}$$

where $b \in \mathbf{R}$ is a bias term representing assumptions about the apriori probability of a signal trial. By sweeping across different values of $b$ the receiver-operator characteristic (ROC) curve is generated. This curve shows how the probability of a hit, $P(D = \text{signal} \mid A)$, and the probability of a false alarm, $P(D = \text{signal} \mid A^c)$, are related. From this curve the parameter $d'$, a measure of sensitivity independent of apriori assumptions, can be estimated. Figure 2 shows the ROC curve as found by numerical simulation, and the curve obtained by the best fit value $d' = 1.89$. This value of $d'$ corresponds to a 82.7% correct detection rate for equal prior signal probabilities.

The theoretically ideal observer can be derived for this problem, since the profile of the unperturbed signal is known exactly (Poor, 1994, p. 278ff). For this problem the optimal observer achieves $d'_{max} = 2$, which implies at equal probabilities for signal and noise trials, the Bayesian limit corresponds to a 84.1% correct detection rate. The detection system based upon the diffusion network is therefore operating close to the Bayesian limit, but was designed using only implicit information, i.e., 100 training examples, about the structure of the signal to be detected, in contrast to the explicit information required to design the optimal Bayesian classifier.

## Footnotes

*To whom correspondence should be addressed.

[1]In our treatment we make no distinction between observables which are inputs and those which are outputs. Inputs can be conceptualized as observables under "environmental control," i.e., whose drifts are independent of both $\lambda$ and the hidden and output processes.

[2]To ease interpretation of (6) consider the simpler case of a one-dimensional Gaussian random variable with mean $\mu$ and variance $\sigma^2$. The ratio of the density of such a model with respect to an equivalent model with zero mean is $\exp(\frac{1}{\sigma^2}\mu x - \frac{1}{2\sigma^2}\mu^2)$. Equation (6) can be viewed as a generalization of this same idea to Brownian motion.

[3]See (Levanony et al., 1990) for sufficient conditions for the differentiation in equation (10) to be valid.
